# Higher Order Statistical Decorrelation without Information Loss

**Gustavo Deco**
Siemens AG
Central Research
Otto-Hahn-Ring 6
81739 Munich
Germany

**Wilfried Brauer**
Technische Universität München
Institut für Informatik
Arcisstr. 21
80290 Munich
Germany

## Abstract

A neural network learning paradigm based on information theory is proposed as a way to perform in an unsupervised fashion, redundancy reduction among the elements of the output layer without loss of information from the sensory input. The model developed performs nonlinear decorrelation up to higher orders of the cumulant tensors and results in probabilistically independent components of the output layer. This means that we don't need to assume Gaussian distribution neither at the input nor at the output. The theory presented is related to the unsupervised-learning theory of Barlow, which proposes redundancy reduction as the goal of cognition. When nonlinear units are used nonlinear principal component analysis is obtained. In this case nonlinear manifolds can be reduced to minimum dimension manifolds. If such units are used the network performs a generalized principal component analysis in the sense that non-Gaussian distributions can be linearly decorrelated and higher orders of the correlation tensors are also taken into account. The basic structure of the architecture involves a general transformation that is volume conserving and therefore the entropy, yielding a map without loss of information. Minimization of the mutual information among the output neurons eliminates the redundancy between the outputs and results in statistical decorrelation of the extracted features. This is known as factorial learning.

# 1  INTRODUCTION

One of the most important theories of feature extraction is the one proposed by Barlow (1989). Barlow describes the process of cognition as a preprocessing of the sensorial information performed by the nervous system in order to extract the statistically relevant and independent features of the inputs without loosing information. This means that the brain should statistically decorrelate the extracted information. As a learning strategy Barlow (1989) formulated the principle of redundancy reduction. This kind of learning is called factorial learning. Recently Atick and Redlich (1992) and Redlich (1993) concentrate on the original idea of Barlow yielding a very interesting formulation of early visual processing and factorial learning. Redlich (1993) reduces redundancy at the input by using a network structure which is a reversible cellular automaton and therefore guarantees the conservation of information in the transformation between input and output. Some nonlinear extensions of PCA for decorrelation of sensorial input signals were recently introduced. These follow very closely Barlow's original ideas of unsupervised learning. Redlich (1993) use similar information theoretic concepts and reversible cellular automata architectures in order to define how nonlinear decorrelation can be performed. The aim of our work is to formulate a neural network architecture and a novel learning paradigm that performs Barlow's unsupervised learning in the most general fashion. The basic idea is to define an architecture that assures perfect transmission without loss of information. Consequently the nonlinear transformation defined by the neural architecture is always bijective. The architecture performs a volume-conserving transformation (determinant of the Jacobian matrix is equal one). As a particular case we can derive the reversible cellular automata architecture proposed by Redlich (1993). The learning paradigm is defined so that the components of the output signal are statistically decorrelated. Due to the fact that the output distribution is not necessarily Gaussian, even if the input is Gaussian, we perform a cumulant expansion of the output distribution and find the rules that should be satisfied by the higher order correlation tensors in order to be decorrelated.

# 2  THEORETICAL FORMALISM

Let us consider an input vector $\vec{x}$ of dimensionality $d$ with components distributed according to the probability distribution $P(\vec{x})$, which is not factorial, i.e. the components of $\vec{x}$ are correlated. The goal of Barlow's unsupervised learning rule is to find a transformation

$$\vec{y} = \vec{F}(\vec{x}) \tag{2.1}$$

such that the components of the output vector $d$-dimensional $\vec{y}$ are statistically decorrelated.

This means that the probability distributions of the components $y_i$ are independent and therefore,

$$P(\vec{y}) = \prod_i^d P(y_i). \tag{2.2}$$

The objective of factorial learning is to find a neural network, which performs the transformation $\vec{F}(\ )$ such that the joint probability distribution $P(\vec{y})$ of the output signals is factorized as in eq. (2.2). In order to implement factorial learning, the information contained in the input should be transferred to the output neurons without loss but, the probability distribution of the output neurons should be statistically decorrelated. Let us now define

these facts from the information theory perspective. The first aspect is to assure the entropy is conserved, i.e.

$$H(\vec{x}) = H(\vec{y}) \tag{2.3}$$

where the symbol $H(\vec{a})$ denotes the entropy of $\vec{a}$ and $H(\vec{a}/\vec{b})$ the conditional entropy of $\vec{a}$ given $\vec{b}$. One way to achieve this goal is to construct an architecture that independently of its synaptic parameters satisfies always eq. (2.3). Thus the architecture will conserve information or entropy. The transmitted entropy satisfies

$$H(\vec{y}) \leq H(\vec{x}) + \int P(\vec{x}) \, ln \, (det \, (\frac{\partial \vec{F}}{\partial \vec{x}})) \, d\vec{x} \tag{2.4}$$

where equality holds only if $\vec{F}$ is bijective, i.e. reversible. Conservation of information and bijectivity is assured if the neural transformation conserves the volume, which mathematically can be expressed by the fact that the Jacobian of the transformation should have determinant unity. In section 3 we formulate an architecture that always conserves the entropy. Let us now concentrate on the main aspect of factorial learning, namely the decorrelation of the output components. Here the problem is to find a volume-conserving transformation that satisfies eq. (2.2). The major problem is that the distribution of the output signal will not necessarily be Gaussian. Therefore it is impossible to use the technique of minimizing the mutual information between the components of the output as done by Redlich (1993). The only way to decorrelate non-Gaussian distributions is to expand the distribution in higher orders of the correlation matrix and impose the independence condition of eq. (2.2). In order to achieve this we propose to use a cumulant expansion of the output distribution. Let us define the Fourier transform of the output distribution,

$$\phi(\vec{K}) = \int d\vec{y} \; e^{i(\vec{K} \cdot \vec{y})} \; P(\vec{y}) \, ; \phi(K_i) = \int dy_i \; e^{i(K_i \cdot y_i)} \; P(y_i) \tag{2.5}$$

The cumulant expansion of a distribution is (Papoulis, 1991)

$$\phi(\vec{K}) = e^{\sum\limits_{n=1}^{\infty} \frac{i^n}{n!} \sum\limits_{i_1, i_2, \dots, i_n}^{n} \aleph_{i_1, i_2, \dots, i_n} K_{i_1} K_{i_2} \cdots K_{i_n}} \qquad \phi(K_i) = e^{\sum\limits_{n=1}^{\infty} \frac{i^n}{n!} \aleph_i^{(n)} K_i^n} \tag{2.6}$$

In the Fourier space the independence condition is given by (Papoulis, 1991)

$$\phi(\vec{K}) = \prod_i \phi(K_i) \tag{2.7}$$

which is equivalent to

$$ln \, (\phi(\vec{K})) = ln \, (\prod_i \phi(K_i)) = \sum_i ln \, (\phi(K_i)) \tag{2.8}$$

Putting eq. (2.8) and the cumulant expansions of eq. (2.6) together, we obtain that in the case of independence the following equality is satisfied

$$\sum\limits_{n=1}^{\infty} \frac{i^n}{n!} \sum\limits_{i_1, i_2, \dots, i_n}^{n} \aleph_{i_1, i_2, \dots, i_n} K_{i_1} K_{i_2} \cdots K_{i_n} = \sum\limits_{i=1}^{d} \sum\limits_{n=1}^{\infty} \frac{i^n}{n!} \aleph_i^{(n)} K_i^n \tag{2.9}$$

In both expansions we will only consider the first four cumulants. After an extra transformation

$$\vec{y}' = \vec{y} - \overline{(\vec{y})} \tag{2.10}$$

to remove the bias $\overline{(\dot{y})}$, we can rewrite eq. (2.9) using the cumulants expression derived in the Papoulius (1991):

$$-\frac{1}{2}\sum_{i,j}K_iK_j\{C_{ij}-C_i^{(2)}\delta_{ij}\} - \frac{i}{6}\sum_{i,j,k}K_iK_jK_k\{C_{ijk}-C_i^{(3)}\delta_{ijk}\}$$

$$+\frac{1}{24}\sum_{i,j,k,l}K_iK_jK_kK_l\{(C_{ijkl}-3C_{ij}C_{kl})-(C_i^{(4)}-3(C_i^{(2)})^2)\delta_{ijkl}\} = 0 \tag{2.11}$$

Equation (2.11) should be satisfied for all values of $\vec{K}$. The multidimensional correlation tensors $C_{i...j}$ and the one-dimensional higher order moments $C_i^{(n)}$ are given by

$$C_{i...j} = \int d\vec{y}'\ P(\vec{y}')\ y'_i...y'_j\ ;\ \ C_i^{(n)} = \int dy'_i\ P(y'_i)\ (y'_i) \tag{2.12}$$

The $\delta_{i...j}$ denotes Kroenecker's delta. Due to the fact that eq. (2.11) should be satisfied for all $\vec{K}$, all coefficients in each summation of eq. (2.11) must be zero. This means that

$$C_{ij} = 0,\ \ if(i \neq j) \tag{2.13}$$

$$C_{ijk} = 0,\ \ if(i \neq j \vee i \neq k) \tag{2.14}$$

$$C_{ijkl} = 0,\ \ if(\{i \neq j \vee i \neq k \vee i \neq l\} \wedge \neg L) \tag{2.15}$$

$$C_{iijj} - C_{ii}C_{jj} = 0,\ \ if(i \neq j). \tag{2.16}$$

In eq. (2.15) $L$ is the logical expression

$$L = \{(i=j \wedge k=l \wedge j \neq k) \vee (i=k \wedge j=l \wedge i \neq j) \vee (i=l \wedge j=k \wedge i \neq j)\}, \tag{2.17}$$

which excludes the cases considered in eq. (2.16). The conditions of independence given by eqs. (2.13-2.16) can be achieved by minimization of the cost function

$$E = \alpha\sum_{i<j}C_{ij}^2 + \beta\sum_{i<j\leq k}C_{ijk}^2 + \gamma\sum_{i<j\leq k\leq l}C_{ijkl}^2 + \delta\sum_{i<j}(C_{iijj}-C_{ii}C_{jj})^2 \tag{2.18}$$

where $\alpha, \beta, \gamma, \delta$ are the inverse of the number of elements in each summation respectively.

In conclusion, minimizing the cost given by eq. (2.18) with a volume-conserving network, we achieve nonlinear decorrelation of non-Gaussian distributions. It is very easy to test wether a factorized probability distribution (eq. 2.2) satisfies the eqs. (2.13-2.16). As a particular case if only second order terms are used in the cumulant expansion, the learning rule reduces to eq. (2.13), which expresses nothing more than the diagonalization of the second order covariance matrix. In this case, by anti-transforming the cumulant expansion of the Fourier transform of the distribution,we obtain a Gaussian distribution. Diagonalization of the covariance matrix decorrelates statistically the components of the output only if we assume a Gaussian distribution of the outputs. In general the distribution of the output is not Gaussian and therefore higher orders of the cumulant expansion should be taken into account, yielding the learning rule conditions eqs. (2.13-2.16) (up to fourth order, generalization to higher orders is straightforward). In the case of Gaussian distribution, minimization of the sum of the variances at each output leads to statistically decorrelation. This fact has a nice information theoretic background namely the minimization of the mutual information between the output components. Statistical independence as expressed in eq. (2.2) is equivalent to (Atick and Redlich, 1992)

$$MH = \sum_j H(y_j) - H(\underset{\sim}{y}) = 0 \tag{2.19}$$

This means that in order to minimize the redundancy at the output we minimize the mutual information between the different components of the output vector. Due to the fact that the volume-conserving structure of the neural network conserves the entropy, the minimization of MH reduces to the minimization of $\sum_j H(y_j)$ .

## 3 VOLUME-CONSERVING ARCHITECTURE AND LEARNING RULE

In this section we define a neural network architecture that is volume-conserving and therefore can be used for the implementation of the learning rules described in the last section. Figure 1.a shows the basic architecture of one layer. The dimensionality of input and output layer is the same and equal to $d$. A similar architecture was proposed by Redlich (1993b) using the theory of reversible cellular automata.

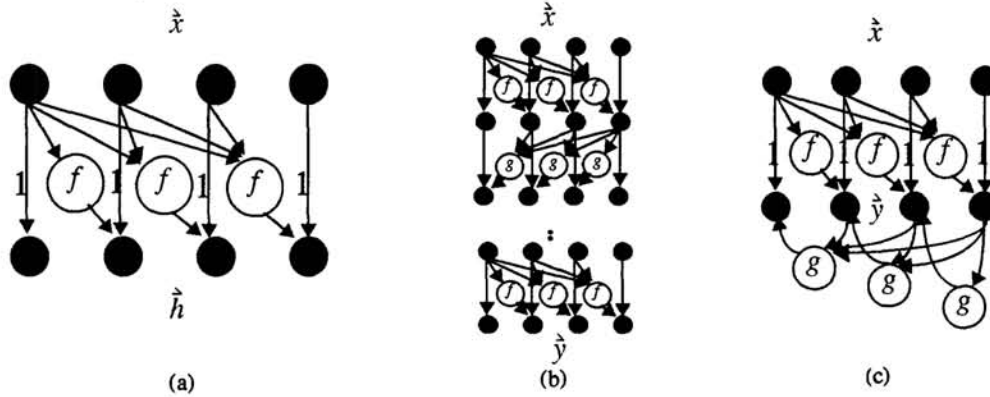

(a)          (b)          (c)

**Figure 1**: Volume-conserving Neural Architectures.

The analytical definition of the transformation defined by this architecture can be written as,

$$y_i = x_i + f_i(x_0, ..., x_j, \vec{\omega}_i). \qquad with \ j < i \qquad (3.1)$$

where $\vec{\omega}_i$ represents a set of parameters of the function $f_i$. Note that independently of the functions $f_i$ the network is always volume-conserving. In particular $f_i$ can be calculated by another neural network, by a sigmoid neuron, by polynomials (higher order neurons), etc. Due to the asymmetric dependence on the input variables and the direct connections with weights equal to 1 between corresponding components of input and output neurons, the Jacobian matrix of the transformation defined in eq. (3.1) is an upper triangular matrix with diagonal elements all equal to one, yielding a determinant equal to one. A network with inverted asymmetry also can be defined as

$$y_i = x_i + g_i(x_j, ..., x_d, \vec{\theta}_i), \qquad with \ i < j \qquad (3.2)$$

corresponding to a lower triangular Jacobian matrix with diagonal elements all equal to one, being therefore volume-conserving. The vectors $\vec{\theta}_i$ represent the parameters of functions $g_i$. In order to yield a general nonlinear transformation from inputs to outputs (without asymmetric dependences) it is possible to build a multilayer architecture like the one shown in Fig. 1.b, which involves mixed versions of networks described by eq. (3.1) and eq. (3.2), respectively. Due to the fact that successive application of volume-conserving transformation is also volume-conserving, the multilayer architecture is also volume-con-

serving. In the two-layer case (Fig. 1.c) the second layer can be interpreted as asymmetric lateral connections between the neurons of the first layer. However, in our case the feed-forward connections between input layer and first layer are also asymmetric. As demonstrated in the last section, we minimize a cost function $E$ to decorrelate nonlinearly correlated non-Gaussian inputs. Let us analyze for simplicity a two-layer architecture (Fig. 1.c) with the first layer given by eq. (3.1) and the second layer by eq. (3.2). Let us denote the output of the hidden layer by $\vec{h}$ and use it as input of eq. (3.2) with output $\vec{y}$. The extension to multilayer architectures is straightforward. The learning rule can be easily expressed by gradient descent method:

$$\vec{\theta}_i = \vec{\theta}_i - \eta \frac{\partial E}{\partial \vec{\theta}_i} \quad ; \quad \vec{\omega}_i = \vec{\omega}_i - \eta \frac{\partial E}{\partial \vec{\omega}_i} \tag{3.3}$$

In order to calculate the derivative of the cost functions we need

$$\frac{C_{i...j}}{\Theta} = \frac{1}{N} \sum_P \{ \frac{\partial}{\partial \Theta}(y_i - \bar{y}_i) ... (y_j - \bar{y}_j) + (y_i - \bar{y}_i) ... \frac{\partial}{\partial \Theta}(y_j - \bar{y}_j ; \quad \frac{\partial \bar{y}_i}{\partial \Theta} = \frac{1}{N} \sum_P \{ \frac{\partial}{\partial \Theta}(y_i) \} \tag{3.4}$$

where $\Theta$ represents the parameters $\vec{\theta}_i$ and $\vec{\omega}_i$. The sums in both equations extend over the $N$ training patterns. The gradients of the different outputs are

$$\frac{\partial}{\partial \vec{\theta}_i} y_i = \frac{\partial}{\partial \vec{\theta}_i} g_i \quad ; \frac{\partial}{\partial \vec{\omega}_i} y_k = (\frac{\partial}{\partial h_i} g_k) (\frac{\partial}{\partial \vec{\omega}_i} f_i) \delta_{i>k} + (\frac{\partial}{\partial \vec{\omega}_i} f_i) \tag{3.5}$$

where $\delta_{i>k}$ is equal to 1 if $i > k$ and 0 otherwise. In this paper we choose a polynomial form for the functions $f$ and $g$. This model involves higher order neurons. In this case each function $f_i$ or $g_i$ is a product of polynomial functions of the inputs. The update equations are given by

$$h_i = \prod_{j=0}^{i-1} \left( \sum_{r=0}^{R} \omega_{ijr} x_j^r \right) \quad ; \quad y_i = \prod_{j=i+1}^{d} \left( \sum_{r=0}^{R} \theta_{ijr} h_j^r \right) \tag{3.6}$$

where $R$ is the order of the polynomial used. In this case the two-layer architecture is a higher order network with a general volume-conserving structure. The derivatives involved in the learning rule are given by

$$\frac{\partial}{\partial \theta_{ijr}} y_k = \delta_{ki} \delta_{j>i} h_j^r \left( \frac{y_i}{\sum_{r=0}^{R} \theta_{ijr} h_j^r} \right) \quad ;$$

$$\frac{\partial}{\partial \omega_{ijr}} y_k = \left( \left( \frac{y_k}{\sum_{r=0}^{R} \theta_{kir} h_i^r} \right) \left( \sum_{r=0}^{R} \theta_{kir} r h_i^{r-1} \right) \delta_{i>k} + \delta_{ik} \right) \delta_{j<i} x_j^r \left( \frac{h_i}{\sum_{r=0}^{R} \omega_{ijr} x_j^r} \right) \tag{3.7}$$

## 4 RESULTS AND SIMULATIONS

We will present herein two different experiments using the architecture defined in this paper. The input space in all experiments is two-dimensional in order to show graphically the results and effects of the presented model. The experiments aim at learning noisy non-linear polynomial and rational curves. Figure 2.a and 2.b plot the input and output space of the second experiment after training is finished, respectively. In this case the noisy logistic map was used to generate the input:

$$x_2 = 4x_1 (1 - x_1) + \upsilon \qquad (4.1)$$

where $\upsilon$ introduces 1% Gaussian noise. In this case a one-layer polynomial network with $R = 2$ was used. The learning constant was $\eta = 0.01$ and 20000 iterations of training were performed. The result of Fig. 2.b is remarkable. The volume-conserving network decorrelated the output space extracting the strong nonlinear correlation that generated the curve in the input space. This means that after training only one coordinate is important to describe the curve.

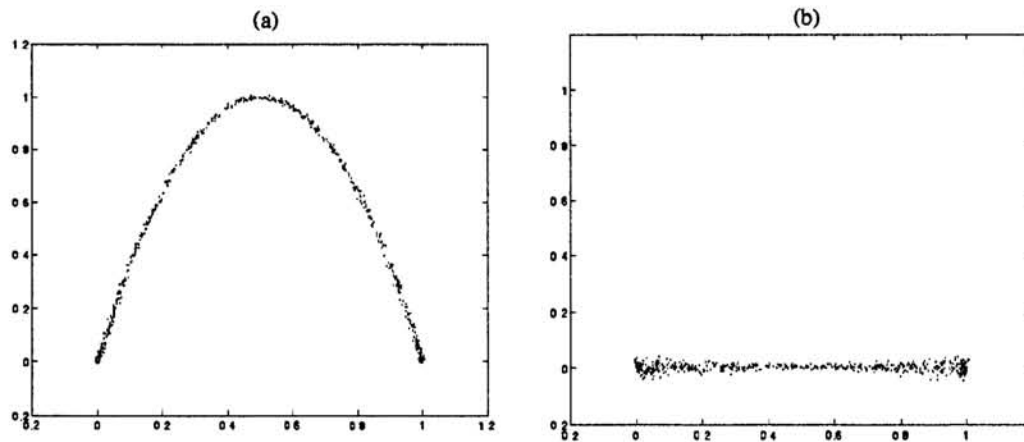

Figure 2: Input and Output space distribution after training with a one-layer polynomial volume-conserving network of order for the logistic map. (a) input space; (b) output space.

The whole information was compressed into the first coordinate of the output. This is the generalization of data compression normally performed by using linear PCA (also called Karhunen-Loewe transformation). The next experiment is similar, but in this case a two-layer network of order $R = 4$ was used. The input space is given by the rational function

$$x_2 = 0.2x_1 + \frac{x_1^3}{(1 + x_1^2)} + \upsilon \qquad (4.2)$$

where $x_1$ and $\upsilon$ are as in the last case. The results are shown in Fig. 4.a (input space) and Fig. 4.b (output space). Fig. 4.c shows the evolution of the four summands of eq (2.18) during learning. It is important to remark that at the beginning the tensors of second and third order are equally important. During learning all summands are simultaneously minimized, resulting in a statistically decorrelated output. The training was performed during 20000 iterations and the learning constant was $\eta = 0.005$ .

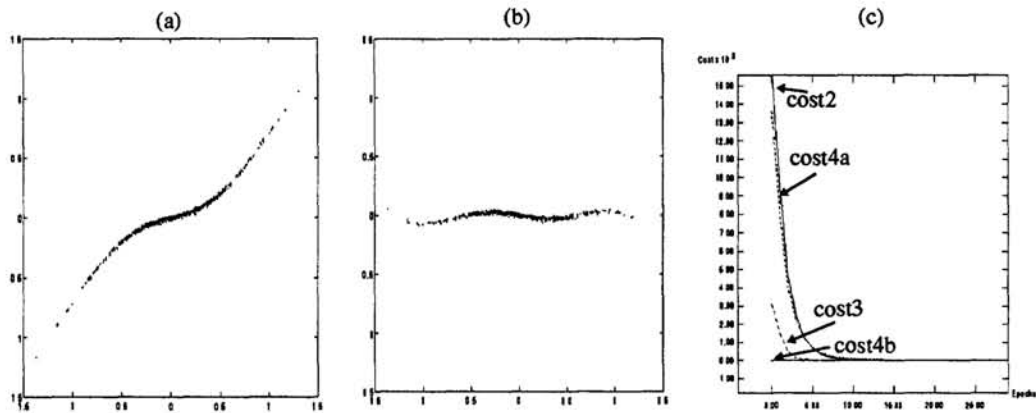

**Figure 4:** Input and Output space distribution after training with a two-layer polynomial volume-conserving network of order  for the noisy curve of eq. (4.2). (a) input space; (b) output space (c) Development of the four summands of the cost function  (eq. 2.18) during learning: (cost 2) first summand (second order correlation tensor); (cost 3) second summand (third correlation order tensor); (cost 4a) third summand (fourth order correlation tensor); (cost4b) fourth summand (fourth order correlation tensor).

## 5  CONCLUSIONS

We proposed a unsupervised neural paradigm, which is based on Information Theory. The algorithm performs redundancy reduction among the elements of the output layer without loosing information, as the data is sent through the network. The model developed performs a generalization of Barlow's unsupervised learning, which consists in nonlinear decorrelation up to higher orders of the cumulant tensors. After training the components of the output layer are statistically independent. Due to the use of higher order cumulant expansion arbitrary non-Gaussian distributions can be rigorously handled. When nonlinear units are used nonlinear principal component analysis is obtained. In this case nonlinear manifolds can be reduced to a minimum dimension manifolds. When linear units are used, the network performs a generalized principal component analysis in the sense that non-Gaussian distribution can be linearly decorrelated.This paper generalizes previous works on factorial learning in two ways: the architecture performs a general nonlinear transformation without loss of information and the decorrelation is performed without assuming Gaussian distributions.

**References:**

H. Barlow. (1989) Unsupervised Learning. *Neural Computation*, 1, 295-311.
A. Papoulis. (1991) *Probability, Random Variables, and Stochastic Processes*. 3. Edition, McGraw-Hill, New York.
A. N. Redlich. (1993) Supervised Factorial Learning. *Neural Computation*, 5, 750-766.